# Competition and Arbors in Ocular Dominance

**Peter Dayan**

Gatsby Computational Neuroscience Unit, UCL
17 Queen Square, London, England, WC1N 3AR.
`dayan@gatsby.ucl.ac.uk`

## Abstract

Hebbian and competitive Hebbian algorithms are almost ubiquitous in modeling pattern formation in cortical development. We analyse in theoretical detail a particular model (adapted from Piepenbrock & Obermayer, 1999) for the development of 1d stripe-like patterns, which places competitive and interactive cortical influences, and free and restricted initial arborisation onto a common footing.

## 1 Introduction

Cats, many species of monkeys, and humans exibit ocular dominance stripes, which are alternating areas of primary visual cortex devoted to input from (the thalamic relay associated with) just one or the other eye (see Erwin *et al*, 1995; Miller, 1996; Swindale, 1996 for reviews of theory and data). These well-known fingerprint patterns have been a seductive target for models of cortical pattern formation because of the mix of competition and cooperation they suggest. A wealth of synaptic adaptation algorithms has been suggested to account for them (and also the concomitant refinement of the topography of the map between the eyes and the cortex), many of which are based on forms of Hebbian learning. Critical issues for the models are the degree of correlation between inputs from the eyes, the nature of the initial arborisation of the axonal inputs, the degree and form of cortical competition, and the nature of synaptic saturation (preventing weights from changing sign or getting too large) and normalisation (allowing cortical and/or thalamic cells to support only a certain total synaptic weight). Different models show different effects of these parameters as to whether ocular dominance should form at all, and, if it does, then what determines the widths of the stripes, which is the main experimental observable.

Although particular classes of models excite fervid criticism from the experimental community, it is to be hoped that the general principles of competitive and cooperative pattern formation that underlie them will remain relevant. To this end we seek models in which we can understand the interactions amongst the various issues above. Piepenbrock & Obermayer (1999) suggested an interesting model in which varying a single parameter spans a spectrum from cortical competition to cooperation. However, the nature of competition in their model makes it hard to predict the outcome of adaptation completely, except in some special cases. In this paper, we suggest a slightly different model of competition which makes the analysis tractable, and simultaneously generalise the model to consider an additional spectrum between flat and peaked arborisation.

## 2 The Model

Figure 1 depicts our model. It is based on the competitive model of Piepenbrock & Obermayer (1999), who developed it in order to explore a continuum between competitive and linear cortical interactions. We use a slightly different competition mechanism and also

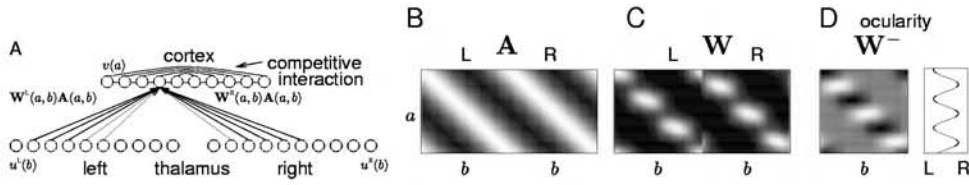

Figure 1: Competitive ocular dominance model. A) Left (L) and right (R) input units (with activities $u^L(b)$ and $u^R(b)$ at the same location $b$ in input space) project through weights $\mathbf{W}^L(a,b)$ and $\mathbf{W}^R(a,b)$ and a restricted topography arbor function $\mathbf{A}(a,b)$ (B) to an output layer, which is subject to lateral competitive interactions. C) Stable weight patterns $\mathbf{W}(a,b)$ showing ocular dominance. D) (left) difference in the connections $\mathbf{W}^- = \mathbf{W}^R - \mathbf{W}^L$ from right and left eye; (right) sum difference across $b$ showing the net ocularity for each $a$. Here, $\sigma_A = 0.2$, $\sigma_I = 0.08$, $\sigma_U = 0.075$, $\beta = 10$, $\gamma = 0.95, \Omega = 3$. There are $N = 100$ units in each input layer and the output layer. Circular (toroidal) boundary conditions are used with $b \in [0, 1)$.

extend the model with an arbor function (as in Miller *et al,* 1989). The model has two input layers (representing input from the thalamus from left 'L' and right 'R' eyes), each containing $N$ units, laid out in a single spatial dimension. These connect to an output layer (layer IV of area V1) with $N$ units too, which is also laid out in a single spatial dimension. We use a continuum approximation, so labeling weights $\mathbf{W}^L(a,b)$ and $\mathbf{W}^R(a,b)$. An *arbor* function, $\mathbf{A}(a,b)$, represents the multiplicity of each such connection (an example is given in figure 1B). The total strengths of the connections from $b$ to $a$ are the products $\mathbf{W}^L(a,b)\mathbf{A}(a,b)$ and $\mathbf{W}^R(a,b)\mathbf{A}(a,b)$.

Four characteristics define the model: the arbor function, the statistics of the input; the mapping from input to output; and the rule by which the weights change. The arbor function $\mathbf{A}(a,b)$ specifies the basic topography of the map at the time that the pattern of synaptic growth is being established. We consider $\mathbf{A}(a,b) \propto e^{-(a-b)^2/2\sigma_A^2}$, where $\sigma_A$ is a parameter specifies its width (figure 1B). The two ends of the spectrum for the arbor are flat, when $\mathbf{A}(a,b) = \alpha$ is constant ($\sigma_A = \infty$), and rigid or punctate, when $\mathbf{A}(a,b) \propto \delta(a-b)$ ($\sigma_A = 0$) and so input cells are mapped only to their topographically matched cells in the cortex.

The second component of the model is the input. Since the model is non-linear, pattern formation is a function of aspects of the input in addition to the two-point correlations between input units that drive development of standard, non-competitive, Hebbian models. We follow Piepenbrock & Obermayer (1999) and consider highly spatially simplified input activities at location $b$ in the left ($u^L(b)$) and right ($u^R(b)$) projections, reflecting just a single Gaussian bump (of width $\sigma_U$) which is stronger to the tune of $\gamma$ in (a randomly chosen) one of the input projections than the other

$$u^L(b) = 0.5(1 + z\gamma)e^{-(b-\xi)^2/2\sigma_U^2} \qquad u^R(b) = 0.5(1 - z\gamma)e^{-(b-\xi)^2/2\sigma_U^2} \qquad (1)$$

where $\xi \in [0, 1)$ is the randomly chosen input location, $z$ is $-1$ or $1$ (with probability 0.5 each), and determines whether the input is more from the right or left projection. $0 \le \gamma \le 1$ governs the weakness of correlations between the projections.

The third component of the model is the way that input activities and the weights conspire to form output activities. This happens in linear (l), competitive (c) and interactive (i) steps:

$$1: \quad v(a) = \int db \, \mathbf{A}(a,b) \left( \mathbf{W}^L(a,b)u^L(b) + \mathbf{W}^R(a,b)u^R(b) \right) , \qquad (2)$$

$$c: \quad v^c(a) = (v(a))^\beta / \int da' \, (v(a'))^\beta \qquad i: \quad v^i(a) = \int da' \, \mathbf{I}(a,a')v^c(a) \qquad (3)$$

Weights, arbor and input and output activities are all positive. In equation 3c, $\beta \ge 1$ is a parameter governing the strength of competition between the cortical cells. As $\beta \to \infty$, the activation process becomes more strongly competitive, ultimately having a winner-takes-all effect as in the standard self-organising map. This form of competition makes it possible

to perform analyses of pattern formation that are hard for the model of Piepenbrock & Obermayer (1999). A natural form for the cortical interactions of equation 3i is the purely positive Gaussian $\mathbf{I}(a, a') = e^{-(a-a')^2/2\sigma_I^2}$.

The fourth component of the model is the weight adaptation rule which involves the Hebbian correlation between input and output activities, averaged over input patterns $\xi z$. The weights are constrained $\mathbf{W}(a, b) \in [0, 1]$, and also multiplicatively normalised so $\int db\, \mathbf{A}(a, b)(\mathbf{W}^L(a, b) + \mathbf{W}^R(a, b)) = \Omega$, for all $a$.

$$\mathbf{W}^L(a, b) \to \mathbf{W}^L(a, b) + \epsilon(\langle v^i(a)u^L(b)\rangle_{\xi z} - \lambda_{(a)}\mathbf{W}^L(a, b)) . \tag{4}$$

(similarly for $\mathbf{W}^R$) where $\lambda_{(a)} = \lambda_{(a)}(\mathbf{W}^L, \mathbf{W}^R)$ is chosen to enforce normalisation.

The initial values for the weights are $\mathbf{W}^{L,R} = \omega e^{-(a-b)^2/2\sigma_W^2} + \eta\delta\mathbf{W}^{L,R}$, where $\omega$ is chosen to satisfy the normalisation constraints, $\eta$ is small, and $\delta\mathbf{W}^L(a, b)$ and $\delta\mathbf{W}^R(a, b)$ are random perturbations constrained so that normalisation is still satisfied. Values of $\sigma_W^2 < \infty$ can emerge as equilibrium values of the weights if there is sufficient competition (sufficiently large $\beta$) or a restricted arbor ($\sigma_A^2 < \infty$).

## 3  Pattern Formation

We analyse pattern formation in the standard manner, finding the equilibrium points (which requires solving a non-linear equation), linearising about them and finding which linear mode grows the fastest. By symmetry, the system separates into two modes, one involving the sum of the weight perturbations $\delta\mathbf{W}^+ = \delta\mathbf{W}^R + \delta\mathbf{W}^L$, which governs the precision of the topography of the final mapping, and one involving the difference $\delta\mathbf{W}^+ = \delta\mathbf{W}^R - \delta\mathbf{W}^L$, which governs ocular dominance. The development of ocular dominance requires that a mode of $\delta\mathbf{W}^-(a, b) \neq 0$ grows, for which each output cell has weights of only one sign (either positive or negative). The stripe width is determined by changes in this sign across the output layer. Figure 1C;D show the sort of patterns for which we would like to account.

**Equilibrium solution**

The equilibrium values of the weights can be found by solving

$$\langle v^i(a)u^L(b)\rangle = \lambda_+ \mathbf{W}^L(a, b) \qquad \langle v^i(a)u^R(b)\rangle = \lambda_+ \mathbf{W}^R(a, b) \tag{5}$$

for the $\lambda_+$ determined such that the normalisation constraint $\int db\, \mathbf{W}^L(a, b) + \mathbf{W}^R(a, b) = \Omega$ is satisfied for all $a$. $v(a)$ is a non-linear function of the weights; however, the simple form of the inputs means that at least one set of equilibrium values of $\mathbf{W}^L(a, b)$ and $\mathbf{W}^R(a, b)$ are the same, $\mathbf{W}^L(a, b) = \omega e^{-(a-b)^2/2\sigma_W^2}$ for a particular width $\sigma_W$ that depends on $I = 1/\sigma_I^2$, $A = 1/\sigma_A^2$, $U = 1/\sigma_U^2$ and $\beta$ according to a simple quadratic equation. We assume that $\omega < 1$, so the weights do not reach their upper saturating limit, and this implies that $\omega = \frac{\Omega}{2N}\sqrt{(A+W)/\pi}$ .

The quadratic equation governing the equilibrium width can be derived by postulating Gaussian weights, and finding the values successively of $v(a), v^q(a)$ and $v^i(a)$ of equations 2 and 3, calculating $\langle\langle v^i(a)u^L(b)\rangle\rangle_{\xi z}$ and finding a consistency condition that $W$ must satisfy in order for $\mathbf{W}^L(a, b) \to \mathbf{W}^L(a, b)$ in equation 4. The result is

$$((\beta + 1)I + \beta U)W^2 + (A((\beta + 1)I + \beta U) - (\beta - 1)UI)W - \beta AIU = 0 \tag{6}$$

Figure 2 shows how the resulting physically realisable ($W > 0$) equilibrium value of $\sigma_W$ depends on $\beta, \sigma_A$ and $\sigma_I$, varying each in turn about a single set of values in figure 1. Figure 2A shows that the width rapidly asymptotes as $\beta$ grows, and it only gets large as the arbor function gets large for $\beta$ near 1. Figure 2B shows this in another way. For $\beta = 1$ (the dotted line), which quite closely parallels the non-competitive case of Miller *et al* (1989),

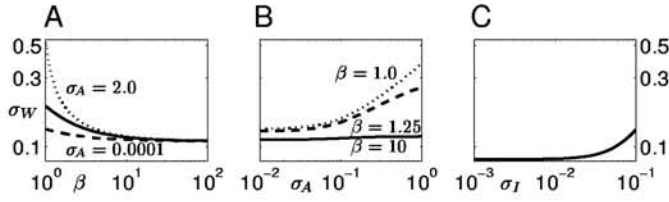

Figure 2: Log-log plots of the equilibrium values of $\sigma_W$ in the case of multiplicative normalisation. Solid lines based on parameters as in figure 1 ($\sigma_A = 0.2$, $\sigma_I = 0.08$, $\sigma_U = 0.075$, $\beta = 10$). A) $\sigma_W$ as a function of $\beta$ for $\sigma_A = 0.2$ (solid), $\sigma_A = 2.0$ (dotted) and $\sigma_A = 0.0001$ (dashed). B) $\sigma_W$ as a function of $\sigma_A$ for $\beta = 10$ (solid), $\beta = 1.25$ (dashed) and $\beta = 1.0$ (dotted). C) $\sigma_W$ as a function of $\sigma_I$. Other parameters as for the solid lines.

$\sigma_W$ grows roughly like the square root of $\sigma_A$ as the arborisation gets flatter. For any $\beta > 1$, one equilibrium value of $\sigma_W$ has a finite asymptote with $\sigma_A$. For absolutely flat topography ($\sigma_A = \infty$) and $\beta > 1$, there are actually two equilibrium values for $\sigma_W$, one with $\sigma_W = \infty$, *ie* flat weights; the other with $\sigma_W$ taking values such as the asymptotic values for the dotted and solid lines in figure 2B.

**The sum mode**

The update equation for (normalised) perturbations to the sum mode is $\delta \mathbf{W}^+(a, b) \rightarrow$

$$(1 - \epsilon\lambda_+)\delta\mathbf{W}^+(a, b) + \epsilon\frac{\beta}{2} \iint da_1 db_1 \, \mathsf{O}(a, b, a_1, b_1)\delta\mathbf{W}^+(a_1, b_1) - \epsilon\lambda'_{(a)}\mathbf{W}^+(a, b) \quad (7)$$

where the operator $\mathsf{O} = \mathsf{O}^1 - \mathsf{O}^2$ is defined by averaging over $\xi$ with $z = 1, \gamma = 1$

$$\mathsf{O}^1(a, b, a_1, b_1) = \left\langle \int da_2 \mathbf{I}(a, a_2)v^q(a_2)\frac{\delta(a_1 - a_2)}{v(a_1)}\mathbf{A}(a_1, b_1)u^R(b_1)u^R(b) \right\rangle \quad (8)$$

$$\mathsf{O}^2(a, b, a_1, b_1) = \left\langle \int da_2 \mathbf{I}(a, a_2)v^q(a_2)\frac{v^q(a_1)}{v(a_1)}\mathbf{A}(a_1, b_1)u^R(b_1)u^R(b) \right\rangle , \quad (9)$$

where, for convenience, we have hidden the dependence of $v(a)$ and $v^q(a)$ on $\xi$ and $z$. Here, the values of $\lambda_+$ and

$$\lambda'_{(a)} = \beta \iiint db\, da_1\, db_1 \, \mathbf{A}(a, b)\mathsf{O}(a, b, a_1, b_1)\delta\mathbf{W}^+(a_1, b_1)/2\Omega \quad (10)$$

come from the normalisation condition. The value of $\lambda_+$ is determined by $\mathbf{W}^+(a, b)$ and not by $\delta\mathbf{W}^+(a_1, b_1)$. Except in the special case that $\sigma_A = \infty$, the term $\epsilon\lambda'_{(a)}\mathbf{W}^+(a, b)$ generally keeps stable the equilibrium solution.

We consider the full eigenfunctions of $\mathsf{O}(a, b, a_1, b_1)$ below. However, the case that Piepenbrock & Obermayer (1999) studied of a flat arbor function ($\sigma_A = \infty$) turns out to be special, admitting two equilibrium solutions, one flat, one with topography, whose stability depends on $\beta$. For $\sigma_A < \infty$, the only Gaussian equilibrium solution for the weights has a refined topography (as one might expect), and this is stable. This width depends on the parameters in a way shown in equation 6 and figure 2, in particular, reaching a non-zero asymptote even as $\beta$ gets very large.

**The difference mode**

The sum mode controls the refinement of topography, whereas the difference mode controls the development and nature of ocular dominance. The equilibrium value of $\mathbf{W}^-(a, b)$ is always $\mathbf{0}$, by symmetry, and the linearised difference equation for the mode is

$$\delta\mathbf{W}^-(a, b) \rightarrow (1 - \epsilon\lambda_+)\delta\mathbf{W}^-(a, b) + \epsilon\frac{\beta\gamma^2}{2} \iint da_1 db_1 \, \mathsf{O}(a, b, a_1, b_1)\delta\mathbf{W}^-(a_1, b_1) \quad (11)$$

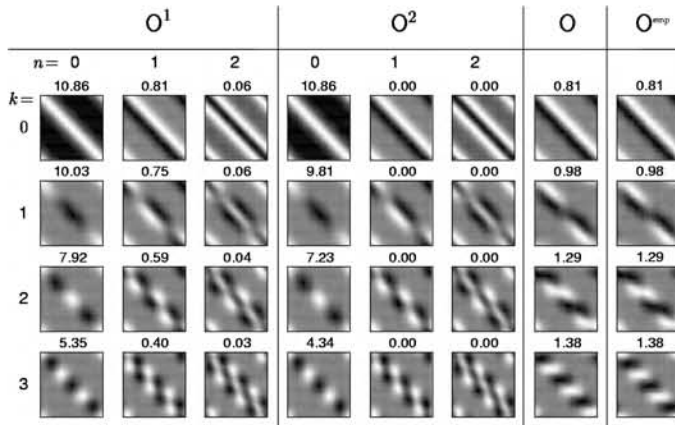

| | $O^1$ | | | $O^2$ | | | $O$ | $O^{emp}$ |
|---|---|---|---|---|---|---|---|---|
| $n=$ | 0 | 1 | 2 | 0 | 1 | 2 | | |
| | 10.86 | 0.81 | 0.06 | 10.86 | 0.00 | 0.00 | 0.81 | 0.81 |
| $k=$ 0 | | | | | | | | |
| | 10.03 | 0.75 | 0.06 | 9.81 | 0.00 | 0.00 | 0.98 | 0.98 |
| 1 | | | | | | | | |
| | 7.92 | 0.59 | 0.04 | 7.23 | 0.00 | 0.00 | 1.29 | 1.29 |
| 2 | | | | | | | | |
| | 5.35 | 0.40 | 0.03 | 4.34 | 0.00 | 0.00 | 1.38 | 1.38 |
| 3 | | | | | | | | |

Figure 3: Eigenfunctions and eigenvalues of $O^1$ (left block), $O^2$ (centre block), and and the theoretical and empirical approximations to $O$ (right columns). Here, as in equation 12, $k$ is the frequency of alternation of ocularity across the output (which is integral for a finite system); $n$ is the order of the Hermite polynomial. The numbers on top of each eigenfunction is the associated eigenvalue. Parameters are as in figure 1 with $\gamma = 1$.

which is almost the same as equation 7 (with the same operator $O$), except that the multiplier for the integral is $\beta\gamma^2/2$ rather than $\beta/2$. Since $\gamma < 1$, the eigenvalues for the difference mode are therefore all less than those for the sum mode, and by the same fraction. The multiplicative decay term $\epsilon\lambda_+\delta\mathbf{W}^-(a,b)$ uses the same $\lambda_+$ as equation 7, whose value is determined exclusively by properties of $\mathbf{W}^+(a,b)$; but the non-multiplicative term $\epsilon\lambda'_{(a)}\mathbf{W}^+(a,b)$ is absent. Note that the equilibrium values of the weights (controlled by $\sigma_W$) affect the operator $O$, and hence its eigenfunctions and eigenvalues.

Provided that the arbor and the initial values of the weights are not both flat ($\sigma_A \neq \infty$ or $\sigma_W \neq \infty$), the principal eigenfunctions of $O^1$ and $O^2$ have the general form

$$\mathbf{W}^-(a,b) = e^{2\pi ika}e^{-d^2(k)(b-a)^2+2\pi il(k)(b-a)}p_n(b-a,k) \qquad (12)$$

where $p_n(r,k)$ is a polynomial (related to a Hermite polynomial) of degree $n$ in $r$ whose coefficients depend on $k$. Here $k$ controls the periodicity in the projective field of each input cell $b$ to the output cells, and ultimately the periodicity of any ocular dominance stripes that might form. The remaining terms control the receptive fields of the output cells. Operator $O^2$ has zero eigenvalues for the polynomials of degree $n > 0$. The expressions for the coefficients of the polynomials and the non-zero eigenvalues of $O^1$ and $O^2$ are rather complicated. Figure 3 shows an example of this analysis. The left $4 \times 3$ block shows eigenfunctions and eigenvalues of $O^1$ for $k = 0\ldots5$ and $n = 0, 1, 2$; the middle $4 \times 3$ block, the equivalent eigenfunctions and eigenvalues of $O^2$. The eigenvalues come essentially from a Gaussian, whose standard deviation is smaller for $O^2$. To a crude first approximation, therefore, the eigenvalues of $O$ resemble the difference of two Gaussians in $k$, and so have a peak at a non-zero value of $k$, ie a finite ocular dominance periodicity.

However, this approximation is too crude. Although the eigenfunctions of $O^1$ and $O^2$ shown in figure 3 *look* almost identical, they are, in fact, subtly different, since $O^1$ and $O^2$ do not commute (except for flat or rigid topography). The similarity between the eigenfunctions makes it possible to approximate the eigenfunctions of $O$ very closely by expanding those of $O^2$ in terms of $O^1$ (or vice-versa). This only requires knowing the overlap between the eigenfunctions, which can be calculated analytically from their form in equation 12. Expanding for $n \leq 2$ leads to the approximate eigenfunctions and eigenvalues for $O$ shown in the penultimate column on the right of figure 3. The difference, for instance, between the

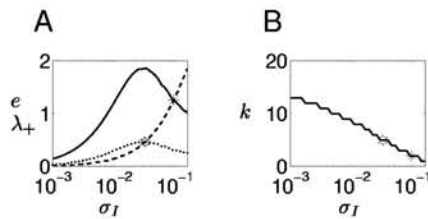

A    B

Figure 4: A) The constraint term $\lambda_+(\Omega/N)$ (dotted line) and the ocular dominance eigenvalues $e(k)(\Omega/N)$ (solid line $\gamma = 1$; dotted line $\gamma = 0.5$) of $\beta\gamma^2 O/2$ as a function of $\sigma_I$, where $k$ is the stripe frequency associated with the maximum eigenvalue. For $\sigma_I$ too large, the ocular dominance eigenfunction no longer dominates. The star and hexagon show the maximum values of $\sigma_I$ such that ocular dominance can form in each case. The scale in (A) is essentially arbitrary. B) Stripe frequency $k$ associated with the largest eigenvalue as a function of $\sigma_I$. The star and hexagon are the same as in (A), showing that the critical preferred stripe frequency is greater for higher correlations between the inputs (lower $\gamma$). Only integer values are considered, hence the apparent aliasing.

eigenfunction of $O$ for $k = 3$ and those for $O^1$ and $O^2$ is striking, considering the similarity between the latter two. For comparison, the farthest right column shows empirically calculated eigenfunctions and eigenvalues of $O$ (using a $50 \times 50$ grid).

Putting $\delta W^-$ back in terms of ocular dominance, we require that eigenmodes of $O$ resembling the modes with $n = 0$ should grow more strongly than the normalisation makes them shrink; and then the value of $k$ associated with the largest eigenvalue will be the stripe frequency that should be expected to dominate. For the parameters of figure 3, the case with $k = 3$ has the largest eigenvalue, and exactly this leads to the outcome of figure 1C;D.

## 4    Results

We can now predict the outcome of development for any set of parameters. First, the analysis of the behavior of the sum mode (including, if necessary, the point about multiple equilibria for flat initial topography) allows a prediction of the equilibrium value of $\sigma_W$, which indicates the degree of topographic refinement. Second, this value of $\sigma_W$ can be used to calculate the value of the normalisation parameter $\lambda_+$ that affects the growth of $\delta W^+$ and $\delta W^-$. There is then a barrier of $2\lambda_+/\beta\gamma^2$ that the eigenvalues of $O$ must surmount for a solution that is not completely binocular to develop. Third, if the peak eigenvalue of $O$ is indeed sufficiently large that ocular dominance develops, then the favored periodicity is set by the value of $k$ associated with this eigenvalue. Of course, if many eigenfunctions have similarly large eigenvalues, then slightly different stripe periodicities may be observed depending on the initial conditions.

The solid line in figure 4A shows the largest eigenvalue of $\beta\gamma^2 O/2$ as a function of the width of the cortical interactions $\sigma_I$, for $\gamma = 1$, the value of $\sigma_W$ specified through the equilibrium analysis, and values of the other parameters as in figure 1. The dashed line shows $\lambda_+$, which comes from the normalisation. The largest value of $\sigma_I$ for which ocular dominance still forms is indicated by the star. For $\gamma = 0.5$, the eigenvalues are reduced by a factor of $\gamma^2 = 0.25$, and so the critical value of $\sigma_I$ (shown by the hexagram) is reduced. Figure 4B shows the frequency of the stripes associated with the largest eigenvalue. The smaller $\sigma_I$, the greater the frequency of the stripes. This line is jagged because only integers are acceptable as stripe frequencies.

Figure 5 shows the consequences of such relationships slightly differently. Some models consider the possibility that $\sigma_I$ might change during development from a large to a small value. If the frequency of the stripes is most strongly determined by the frequency that grows fastest when $\sigma_I$ is first sufficiently small that stripes grow, we can analyse plots such as those in figure 4 to determine the outcome of development. The figures in the top row

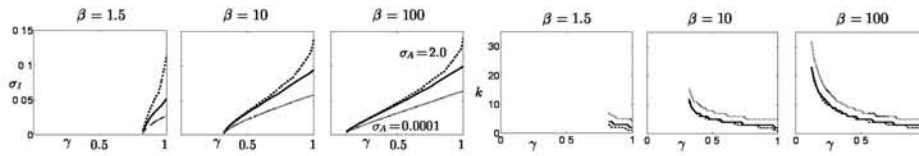

Figure 5: First three figures: maximal values of $\sigma_I$ for which ocular dominance will develop as a function of $\gamma$. All other parameters as in figure 1, except that $\sigma_A = 0.2$ (solid), $\sigma_A = 2.0$ (dashed); $\sigma_A = 0.0001$ (dotted). Last three figures: value of stripe frequency $k$ associated with the maximal eigenvalue for parameters as in the left three plots at the critical value of $\sigma_I$.

show the largest values of $\sigma_I$ for which ocular dominance can develop; the bottom plots show the stripe frequencies associated with these critical values of $\sigma_I$ (like the stars and hexagons in figure 4), in both cases as a function of $\gamma$. The columns are for successively larger values of $\beta$; within each plot there are three lines, for $\sigma_A = 0.0001$ (dotted); $\sigma_A = 0.2$ (solid), and $\sigma_A = 2.0$ (dashed). Where no value of $\sigma_I$ permits ocular dominance to form, no line is shown. From the plots, we can see that the more similar the inputs, (the smaller $\gamma$) or the less the competition (the smaller $\beta$), the harder it is for ocular dominance to form. However, if ocular dominance does form, then the width of the stripes depends only weakly on the degree of competition, and slightly more strongly on the width of the arbors. The narrower the arbor, the larger the frequency of the stripes. For rigid topography, as $\sigma_A \to 0$, the critical value of $\sigma_I$ depends roughly linearly on $\gamma$. We analyse this case in more detail below. Note that the stripe width predicted by the linear analysis does not depend on the correlation between the input projections *unless* other parameters (such as $\sigma_I$) change, although ocular dominance might not develop for some values of the parameters.

## 5   Discussion

The analytical tractability of the model makes it possible to understand in depth the interaction between cooperation, competition, correlation and arborisation. Further exploration of this complex space of interactions is obviously required. Simulations across a range of parameters have shown that the analysis makes correct predictions, although we have only analysed linear pattern formation. Non-linear stability turns out to play a highly significant role in higher dimensions (such as the 2d ocular dominance stripe pattern) where a continuum of eigenmodes share the same eigenvalues (Bressloff & Cowan, personal communication), and also in 1d models involving very strong competition ($\beta \to \infty$) like the self-organising map (Kohonen, 1995).

### Acknowledgements

Funded by the Gatsby Charitable Foundation. I am very grateful to Larry Abbott, Ed Erwin, Geoff Goodhill, John Hertz, Ken Miller, Klaus Obermayer, Read Montague, Nick Swindale, Peter Wiesing and David Willshaw for discussions and to Zhaoping Li for making this paper possible.

### References

Erwin, E, Obermayer, K & Schulten, K (1995) *Neural Computation* 7:425-468.

Kohonen, T (1995) *Self-Organizing Maps*. Berlin, New York:Springer-Verlag.

Miller, KD (1996) In E Domany, JL van Hemmen & K Schulten, eds, *Models of Neural Networks, III*. New York:Springer-Verlag, 55-78.

Miller, KD, Keller, JB & Stryker, MP (1989) *Science* **245**:605-615.

Piepenbrock, C & Obermayer, K (1999). In MS Kearns, SA Solla & DA Cohn, eds, *Advances in Neural Information Processing Systems, 11*. Cambridge, MA: MIT Press.

Swindale, NV (1996) *Network: Computation in Neural Systems* 7:161-247.
